# Reward Design via Online Gradient Ascent

**Jonathan Sorg**
Computer Science and Eng.
University of Michigan
jdsorg@umich.edu

**Satinder Singh**
Computer Science and Eng.
University of Michigan
baveja@umich.edu

**Richard L. Lewis**
Department of Psychology
University of Michigan
rickl@umich.edu

## Abstract

Recent work has demonstrated that when artificial agents are limited in their ability to achieve their goals, the agent designer can benefit by making the agent's goals different from the designer's. This gives rise to the optimization problem of designing the artificial agent's goals—in the RL framework, designing the agent's reward function. Existing attempts at solving this optimal reward problem do not leverage experience gained online during the agent's lifetime nor do they take advantage of knowledge about the agent's structure. In this work, we develop a gradient ascent approach with formal convergence guarantees for approximately solving the optimal reward problem online during an agent's lifetime. We show that our method generalizes a standard policy gradient approach, and we demonstrate its ability to improve reward functions in agents with various forms of limitations.

## 1 The Optimal Reward Problem

In this work, we consider the scenario of an agent designer building an autonomous agent. The designer has his or her own goals which must be translated into goals for the autonomous agent. We represent goals using the Reinforcement Learning (RL) formalism of the reward function. This leads to the *optimal reward problem* of designing the agent's reward function so as to maximize the objective reward received by the agent designer.

Typically, the designer assigns his or her own reward to the agent. However, there is ample work which demonstrates the benefit of assigning reward which does not match the designer's. For example, work on reward shaping [11] has shown how to modify rewards to accelerate learning without altering the optimal policy, and PAC-MDP methods [5, 20] including approximate Bayesian methods [7, 19] add bonuses to the objective reward to achieve optimism under uncertainty. These approaches explicitly or implicitly assume that the asymptotic behavior of the agent should be the same as that which would occur using the objective reward function. These methods do not explicitly consider the optimal reward problem; however, they do show improved performance through reward modification. In our recent work that does explicitly consider the optimal reward problem [18], we analyzed an explicit hypothesis about the benefit of reward design—that it helps mitigate the performance loss caused by computational constraints (bounds) on agent architectures. We considered various types of agent limitations—limits on planning depth, failure to account for partial observability, and other erroneous modeling assumptions—and demonstrated the benefits of good reward functions in each case empirically. Crucially, in bounded agents, the optimal reward function often leads to behavior that is different from the asymptotic behavior achieved with the objective reward function.

In this work, we develop an algorithm, Policy Gradient for Reward Design (PGRD), for improving reward functions for a family of bounded agents that behave according to repeated local (from the current state) model-based planning. We show that this algorithm is capable of improving the reward functions in agents with computational limitations necessitating small bounds on the depth of planning, and also from the use of an inaccurate model (which may be inaccurate due to computationally-motivated approximations). PGRD has few parameters, improves the reward

function online during an agent's lifetime, takes advantage of knowledge about the agent's structure (through the gradient computation), and is linear in the number of reward function parameters.

**Notation.** Formally, we consider discrete-time partially-observable environments with a finite number of hidden states $s \in S$, actions $a \in A$, and observations $o \in O$; these finite set assumptions are useful for our theorems, but our algorithm can handle infinite sets in practice. Its dynamics are governed by a state-transition function $P(s'|s, a)$ that defines a distribution over next-states $s'$ conditioned on current state $s$ and action $a$, and an observation function $\Omega(o|s)$ that defines a distribution over observations $o$ conditioned on current state $s$.

The agent designer's goals are specified via the *objective reward function* $R_\mathcal{O}$. At each time step, the designer receives reward $R_\mathcal{O}(s_t) \in [0, 1]$ based on the current state $s_t$ of the environment, where the subscript denotes time. The designer's *objective return* is the expected mean objective reward obtained over an infinite horizon, i.e., $\lim_{N \to \infty} \mathbb{E}\left[\frac{1}{N} \sum_{t=0}^{N} R_\mathcal{O}(s_t)\right]$. In the standard view of RL, the agent uses the same reward function as the designer to align the interests of the agent and the designer. Here we allow for a separate agent reward function $R(\cdot)$. An agent's reward function can in general be defined in terms of the history of actions and observations, but is often more pragmatically defined in terms of some abstraction of history. We define the agent's reward function precisely in Section 2.

**Optimal Reward Problem.** An RL agent attempts to act so as to maximize its own cumulative reward, or return. Crucially, as a result, the sequence of environment-states $\{s_t\}_{t=0}^{\infty}$ is affected by the choice of reward function; therefore, the agent designer's return is affected as well. The optimal reward problem arises from the fact that while the objective reward function is fixed as part of the problem description, the reward function is a choice to be made by the designer. We capture this choice abstractly by letting the reward be parameterized by some vector of parameters $\theta$ chosen from space of parameters $\Theta$. Each $\theta \in \Theta$ specifies a reward function $R(\cdot; \theta)$ which in turn produces a distribution over environment state sequences via whatever RL method the agent uses. The expected return obtained by the designer for choice $\theta$ is $\mathcal{U}(\theta) = \lim_{N \to \infty} \mathbb{E}\left[\frac{1}{N} \sum_{t=0}^{N} R_\mathcal{O}(s_t) \middle| R(\cdot; \theta)\right]$. The optimal reward parameters are given by the solution to the optimal reward problem [16, 17, 18]:

$$\theta^* = \arg\max_{\theta \in \Theta} \mathcal{U}(\theta) = \arg\max_{\theta \in \Theta} \lim_{N \to \infty} \mathbb{E}\left[\frac{1}{N} \sum_{t=0}^{N} R_\mathcal{O}(s_t) \middle| R(\cdot; \theta)\right]. \tag{1}$$

Our previous research on solving the optimal reward problem has focused primarily on the properties of the optimal reward function and its correspondence to the agent architecture and the environment [16, 17, 18]. This work has used inefficient exhaustive search methods for finding good approximations to $\theta^*$ (though there is recent work on using genetic algorithms to do this [6, 9, 12]). Our primary contribution in this paper is a new convergent online stochastic gradient method for finding approximately optimal reward functions. To our knowledge, this is the first algorithm that improves reward functions in an online setting—during a single agent's lifetime.

In Section 2, we present the PGRD algorithm, prove its convergence, and relate it to OLPOMDP [2], a policy gradient algorithm. In Section 3, we present experiments demonstrating PGRD's ability to approximately solve the optimal reward problem online.

## 2 PGRD: Policy Gradient for Reward Design

PGRD builds on the following insight: the agent's planning algorithm procedurally converts the reward function into behavior; thus, the reward function can be viewed as a specific parameterization of the agent's policy. Using this insight, PGRD updates the reward parameters by estimating the gradient of the objective return with respect to the reward parameters, $\nabla_\theta \mathcal{U}(\theta)$, from experience, using standard policy gradient techniques. In fact, we show that PGRD can be viewed as an (independently interesting) generalization of the policy gradient method OLPOMDP [2]. Specifically, we show that OLPOMDP is special case of PGRD when the planning depth $d$ is zero. In this section, we first present the family of local planning agents for which PGRD improves the reward function. Next, we develop PGRD and prove its convergence. Finally, we show that PGRD generalizes OLPOMDP and discuss how adding planning to OLPOMDP affects the space of policies available to the optimization method.

**Input**: $T, \theta_0, \{\alpha_t\}_{t=0}^{\infty}, \beta, \gamma$

```
1  o_0, i_0 = initializeStart();
2  for t = 0, 1, 2, 3, … do
3  │   ∀_a Q_t(a; θ_t) = plan(i_t, o_t, T, R(i_t, ·, ·; θ_t), d, γ);
4  │   a_t ~ μ(a|i_t; Q_t);
5  │   r_{t+1}, o_{t+1} = takeAction(a_t);
6  │   z_{t+1} = β z_t + ∇_{θ_t} μ(a_t|i_t; Q_t) / μ(a_t|i_t; Q_t);
7  │   θ_{t+1} = θ_t + α_t(r_{t+1} z_{t+1} − λθ_t);
8  │   i_{t+1} = updateInternalState(i_t, a_t, o_{t+1});
9  end
```

Figure 1: PGRD (Policy Gradient for Reward Design) Algorithm

**A Family of Limited Agents with Internal State.** Given a Markov model $T$ defined over the observation space $O$ and action space $A$, denote $T(o'|o, a)$ the probability of next observation $o'$ given that the agent takes action $a$ after observing $o$. Our agents use the model $T$ to plan. We do not assume that the model $T$ is an accurate model of the environment. The use of an incorrect model is one type of agent limitation we examine in our experiments. In general, agents can use non-Markov models defined in terms of the history of observations and actions; we leave this for future work.

The agent maintains an internal state feature vector $i_t$ that is updated at each time step using $i_{t+1} = $ updateInternalState$(i_t, a_t, o_{t+1})$. The internal state allows the agent to use reward functions that depend on the agent's history. We consider rewards of the form $R(i_t, o, a; \theta_t) = \theta_t^T \phi(i_t, o, a)$, where $\theta_t$ is the reward parameter vector at time $t$, and $\phi(i_t, o, a)$ is a vector of features based on internal state $i_t$, planning state $o$, and action $a$. Note that if $\phi$ is a vector of binary indicator features, this representation allows for arbitrary reward functions and thus the representation is completely general. Many existing methods use reward functions that depend on history. Reward functions based on empirical counts of observations, as in PAC-MDP approaches [5, 20], provide some examples; see [14, 15, 13] for others. We present a concrete example in our empirical section.

At each time step $t$, the agent's planning algorithm, plan, performs depth-$d$ planning using the model $T$ and reward function $R(i_t, o, a; \theta_t)$ with current internal state $i_t$ and reward parameters $\theta_t$. Specifically, the agent computes a $d$-step Q-value function $Q^d(i_t, o_t, a; \theta_t) \forall a \in A$, where $Q^d(i_t, o, a; \theta_t) = R(i_t, o, a; \theta_t) + \gamma \sum_{o' \in O} T(o'|o, a) \max_{b \in A} Q^{d-1}(i_t, o', b; \theta_t)$ and $Q^0(i_t, o, a; \theta_t) = R(i_t, o, a; \theta_t)$. We emphasize that the internal state $i_t$ and reward parameters $\theta_t$ are held invariant while planning. Note that the $d$-step Q-values are only computed for the current observation $o_t$, in effect by building a depth-$d$ tree rooted at $o_t$. In the $d = 0$ special case, the planning procedure completely ignores the model $T$ and returns $Q^0(i_t, o_t, a; \theta_t) = R(i_t, o_t, a; \theta_t)$. Regardless of the value of $d$, we treat the end result of planning as providing a scoring function $Q_t(a; \theta_t)$ where the dependence on $d$, $i_t$ and $o_t$ is dropped from the notation. To allow for gradient calculations, our agents act according to the Boltzmann (soft-max) stochastic policy parameterized by $Q$: $\mu(a|i_t; Q_t) \stackrel{\text{def}}{=} \frac{e^{\tau Q_t(a; \theta_t)}}{\sum_b e^{\tau Q_t(b; \theta_t)}}$, where $\tau$ is a temperature parameter that determines how stochastically the agent selects the action with the highest score. When the planning depth $d$ is small due to computational limitations, the agent cannot account for events beyond the planning depth. We examine this limitation in our experiments.

**Gradient Ascent.** To develop a gradient algorithm for improving the reward function, we need to compute the gradient of the objective return with respect to $\theta$: $\nabla_\theta \mathcal{U}(\theta)$. The main insight is to break the gradient calculation into the calculation of two gradients. The first is the gradient of the objective return with respect to the policy $\mu$, and the second is the gradient of the policy with respect to the reward function parameters $\theta$. The first gradient is exactly what is computed in standard policy gradient approaches [2]. The second gradient is challenging because the transformation from reward parameters to policy involves a model-based planning procedure. We draw from the work of Neu and Szepesvári [10] which shows that this gradient computation resembles planning itself. We develop PGRD, presented in Figure 1, explicitly as a generalization of OLPOMDP, a policy gradient algorithm developed by Bartlett and Baxter [2], because of its foundational simplicity relative to other policy-gradient algorithms such as those based on actor-critic methods (e.g., [4]). Notably, the reward parameters are the only parameters being learned in PGRD.

PGRD follows the form of OLPOMDP (Algorithm 1 in Bartlett and Baxter [2]) but generalizes it in three places. In Figure 1 line 3, the agent plans to compute the policy, rather than storing the policy directly. In line 6, the gradient of the policy with respect to the parameters accounts for the planning procedure. In line 8, the agent maintains a general notion of internal state that allows for richer parameterization of policies than typically considered (similar to Aberdeen and Baxter [1]). The algorithm takes as parameters a sequence of learning rates $\{\alpha_k\}$, a decaying-average parameter $\beta$, and regularization parameter $\lambda > 0$ which keeps the the reward parameters $\theta$ bounded throughout learning. Given a sequence of calculations of the gradient of the policy with respect to the parameters, $\nabla_{\theta_t}\mu(a_t|i_t; Q_t)$, the remainder of the algorithm climbs the gradient of objective return $\nabla_\theta\mathcal{U}(\theta)$ using OLPOMDP machinery. In the next subsection, we discuss how to compute $\nabla_{\theta_t}\mu(a_t|i_t; Q_t)$.

**Computing the Gradient of the Policy with respect to Reward.** For the Boltzmann distribution, the gradient of the policy with respect to the reward parameters is given by the equation $\nabla_{\theta_t}\mu(a|i_t; Q_t) = \tau \cdot \mu(a|Q_t)[\nabla_{\theta_t}Q_t(a|i_t;\theta_t) - \sum_{b \in A}\nabla_{\theta_t}Q_t(b;\theta_t)]$, where $\tau$ is the Boltzmann temperature (see [10]). Thus, computing $\nabla_{\theta_t}\mu(a|i_t; Q_t)$ reduces to computing $\nabla_{\theta_t}Q_t(a;\theta_t)$.

The value of $Q_t$ depends on the reward parameters $\theta_t$, the model, and the planning depth. However, as we present below, the process of computing the gradient closely resembles the process of planning itself, and the two computations can be interleaved. Theorem 1 presented below is an adaptation of Proposition 4 from Neu and Szepesvári [10]. It presents the gradient computation for depth-$d$ planning as well as for infinite-depth discounted planning. We assume that the gradient of the reward function with respect to the parameters is bounded: $\sup_{\theta,o,i,a}\|\nabla_\theta R(i, o, a, \theta)\| < \infty$. The proof of the theorem follows directly from Proposition 4 of Neu and Szepesvári [10].

**Theorem 1.** *Except on a set of measure zero, for any depth $d$, the gradient $\nabla_\theta Q^d(o, a; \theta)$ exists and is given by the recursion (where we have dropped the dependence on $i$ for simplicity)*

$$\nabla_\theta Q^d(o, a; \theta) = \nabla_\theta R(o, a; \theta) + \gamma \sum_{o' \in O} T(o'|o, a) \sum_{b \in A} \pi^{d-1}(b|o')\nabla_\theta Q^{d-1}(o', b; \theta), \quad (2)$$

*where $\nabla_\theta Q^0(o, a; \theta) = \nabla_\theta R(o, a; \theta)$ and $\pi^d(a|o) \in \arg\max_a Q^d(o, a; \theta)$ is any policy that is greedy with respect to $Q^d$. The result also holds for $\nabla_\theta Q^*(o, a; \theta) = \nabla_\theta \lim_{d \to \infty} Q^d(o, a; \theta)$.*

The Q-function will not be differentiable when there are multiple optimal policies. This is reflected in the arbitrary choice of $\pi$ in the gradient calculation. However, it was shown by Neu and Szepesvári [10] that even for values of $\theta$ which are not differentiable, the above computation produces a valid calculation of a subgradient; we discuss this below in our proof of convergence of PGRD.

**Convergence of PGRD (Figure 1).** Given a particular fixed reward function $R(\cdot; \theta)$, transition model $T$, and planning depth, there is a corresponding fixed randomized policy $\mu(a|i; \theta)$—where we have explicitly represented the reward's dependence on the internal state vector $i$ in the policy parameterization and dropped $Q$ from the notation as it is redundant given that everything else is fixed. Denote the agent's internal-state update as a (usually deterministic) distribution $\psi(i'|i, a, o)$. Given a fixed reward parameter vector $\theta$, the joint environment-state–internal-state transitions can be modeled as a Markov chain with a $|S||I| \times |S||I|$ transition matrix $M(\theta)$ whose entries are given by $M_{\langle s,i\rangle, \langle s',i'\rangle}(\theta) = p(\langle s', i'\rangle|\langle s, i\rangle; \theta) = \sum_{o,a}\psi(i'|i, a, o)\Omega(o|s')P(s'|s, a)\mu(a|i; \theta)$. We make the following assumptions about the agent and the environment:

**Assumption 1.** *The transition matrix $M(\theta)$ of the joint environment-state–internal-state Markov chain has a unique stationary distribution $\pi(\theta) = [\pi_{s_1,i_1}(\theta), \pi_{s_2,i_2}(\theta), \ldots, \pi_{s_{|S|},i_{|I|}}(\theta)]$ satisfying the balance equations $\pi(\theta)M(\theta) = \pi(\theta)$, for all $\theta \in \Theta$.*

**Assumption 2.** *During its execution, PGRD (Figure 1) does not reach a value of $i_t$, and $\theta_t$ at which $\mu(a_t|i_t, Q_t)$ is not differentiable with respect to $\theta_t$.*

It follows from Assumption 1 that the objective return, $\mathcal{U}(\theta)$, is independent of the start state. The original OLPOMDP convergence proof [2] has a similar condition that only considers environment states. Intuitively, this condition allows PGRD to handle history-dependence of a reward function in the same manner that it handles partial observability in an environment. Assumption 2 accounts for the fact that a planning algorithm may not be fully differentiable everywhere. However, Theorem 1 showed that infinite and bounded-depth planning is differentiable almost everywhere (in a measure theoretic sense). Furthermore, this assumption is perhaps stronger than necessary, as stochastic approximation algorithms, which provide the theory upon which OLPOMDP is based, have been shown to converge using subgradients [8].

In order to state the convergence theorem, we must define the approximate gradient which OLPOMDP calculates. Let the approximate gradient estimate be $\widetilde{\nabla}_\theta^\beta \mathcal{U}(\theta) \stackrel{\text{def}}{=} \lim_{T \to \infty} \sum_{t=1}^{T} r_t z_t$ for a fixed $\theta$ and PGRD parameter $\beta$, where $z_t$ (in Figure 1) represents a time-decaying average of the $\nabla_{\theta_t} \mu(a_t | i_t, Q_t)$ calculations. It was shown by Bartlett and Baxter [2] that $\widetilde{\nabla}_\theta^\beta \mathcal{U}(\theta)$ is close to the true value $\nabla_\theta \mathcal{U}(\theta)$ for large values of $\beta$. Theorem 2 proves that PGRD converges to a stable equilibrium point based on this approximate gradient measure. This equilibrium point will typically correspond to some local optimum in the return function $\mathcal{U}(\theta)$. Given our development and assumptions, the theorem is a straightforward extension of Theorem 6 from Bartlett and Baxter [2] (proof omitted).

**Theorem 2.** *Given $\beta \in [0, 1)$, $\lambda > 0$, and a sequence of step sizes $\alpha_t$ satisfying $\sum_{t=0}^{\infty} \alpha_t = \infty$ and $\sum_{t=0}^{\infty} (\alpha_t)^2 < \infty$, PGRD produces a sequence of reward parameters $\theta_t$ such that $\theta_t \to L$ as $t \to \infty$ a.s., where $L$ is the set of stable equilibrium points of the differential equation $\frac{\partial \theta}{\partial t} = \widetilde{\nabla}_\theta^\beta \mathcal{U}(\theta) - \lambda \theta$.*

**PGRD generalizes OLPOMDP.** As stated above, OLPOMDP, when it uses a Boltzmann distribution in its policy representation (a common case), is a special case of PGRD when the planning depth is zero. First, notice that in the case of depth-0 planning, $Q^0(i, o, a; \theta) = R(i, o, a, \theta)$, regardless of the transition model and reward parameterization. We can also see from Theorem 1 that $\nabla_\theta Q^0(i, o, a; \theta) = \nabla_\theta R(i, o, a; \theta)$. Because $R(i, o, a; \theta)$ can be parameterized arbitrarily, PGRD can be configured to match standard OLPOMDP with any policy parameterization that also computes a score function for the Boltzmann distribution.

In our experiments, we demonstrate that choosing a planning depth $d > 0$ can be beneficial over using OLPOMDP ($d = 0$). In the remainder of this section, we show theoretically that choosing $d > 0$ does not hurt in the sense that it does not reduce the space of policies available to the policy gradient method. Specifically, we show that when using an expressive enough reward parameterization, PGRD's space of policies is not restricted relative to OLPOMDP's space of policies. We prove the result for infinite planning, but the extension to depth-limited planning is straightforward.

**Theorem 3.** *There exists a reward parameterization such that, for an arbitrary transition model $T$, the space of policies representable by PGRD with infinite planning is identical to the space of policies representable by PGRD with depth 0 planning.*

*Proof.* Ignoring internal state for now (holding it constant), let $C(o, a)$ be an arbitrary reward function used by PGRD with depth 0 planning. Let $R(o, a; \theta)$ be a reward function for PGRD with infinite ($d = \infty$) planning. The depth-$\infty$ agent uses the planning result $Q^*(o, a; \theta)$ to act, while the depth-0 agent uses the function $C(o, a)$ to act. Therefore, it suffices to show that one can always choose $\theta$ such that the planning solution $Q^*(o, a; \theta)$ equals $C(o, a)$. For all $o \in O, a \in A$, set $R(o, a; \theta) = C(o, a) - \gamma \sum_{o'} T(o'|o, a) \max_{a'} C(o', a')$. Substituting $Q^*$ for $C$, this is the Bellman optimality equation [22] for infinite-horizon planning. Setting $R(o, a; \theta)$ as above is possible if it is parameterized by a table with an entry for each observation–action pair. $\qquad\square$

Theorem 3 also shows that the effect of an arbitrarily poor model can be overcome with a good choice of reward function. This is because a Boltzmann distribution can, allowing for an arbitrary scoring function $C$, represent any policy. We demonstrate this ability of PGRD in our experiments.

## 3   Experiments

The primary objective of our experiments is to demonstrate that PGRD is able to use experience online to improve the reward function parameters, thereby improving the agent's obtained objective return. Specifically, we compare the objective return achieved by PGRD to the objective return achieved by PGRD with the reward adaptation turned off. In both cases, the reward function is initialized to the objective reward function. A secondary objective is to demonstrate that when a good model is available, adding the ability to plan—even for small depths—improves performance relative to the baseline algorithm of OLPOMDP (or equivalently PGRD with depth $d = 0$).

*Foraging Domain for Experiments 1 to 3:* The foraging environment illustrated in Figure 2(a) is a $3 \times 3$ grid world with 3 dead-end corridors (rows) separated by impassable walls. The agent (bird) has four available actions corresponding to each cardinal direction. Movement in the intended direction fails with probability $0.1$, resulting in movement in a random direction. If the resulting direction is

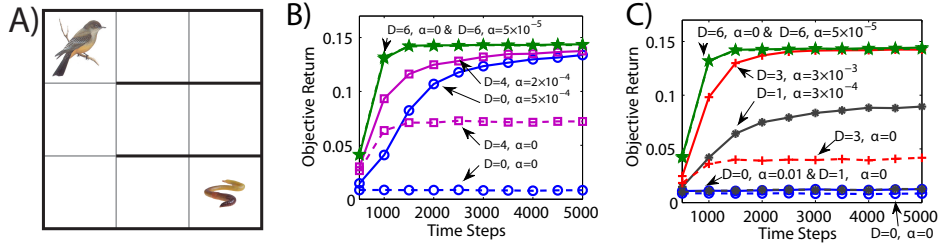

Figure 2: A) Foraging Domain, B) Performance of PGRD with observation-action reward features, C) Performance of PGRD with recency reward features

blocked by a wall or the boundary, the action results in no movement. There is a food source (worm) located in one of the three right-most locations at the end of each corridor. The agent has an `eat` action, which consumes the worm when the agent is at the worm's location. After the agent consumes the worm, a new worm appears randomly in one of the other two potential worm locations.

*Objective Reward for the Foraging Domain:* The designer's goal is to maximize the average number of worms eaten per time step. Thus, the objective reward function $R_\mathcal{O}$ provides a reward of $1.0$ when the agent eats a worm, and a reward of $0$ otherwise. The objective return is defined as in Equation (1).

*Experimental Methodology:* We tested PGRD for depth-limited planning agents of depths 0–6. Recall that PGRD for the agent with planning depth $0$ is the OLPOMDP algorithm. For each depth, we jointly optimized over the PGRD algorithm parameters, $\alpha$ and $\beta$ (we use a fixed $\alpha$ throughout learning). We tested values for $\alpha$ on an approximate logarithmic scale in the range $(10^{-6}, 10^{-2})$ as well as the special value of $\alpha = 0$, which corresponds to an agent that does not adapt its reward function. We tested $\beta$ values in the set $0, 0.4, 0.7, 0.9, 0.95, 0.99$. Following common practice [3], we set the $\lambda$ parameter to 0. We explicitly bound the reward parameters and capped the reward function output both to the range $[-1, 1]$. We used a Boltzmann temperature parameter of $\tau = 100$ and planning discount factor $\gamma = 0.95$. Because we initialized $\theta$ so that the initial reward function was the objective reward function, PGRD with $\alpha = 0$ was equivalent to a standard depth-limited planning agent.

**Experiment 1: A fully observable environment with a correct model learned online.** In this experiment, we improve the reward function in an agent whose only limitation is planning depth, using (1) a general reward parameterization based on the current observation and (2) a more compact reward parameterization which also depends on the history of observations.

*Observation:* The agent observes the full state, which is given by the pair $o = (l, w)$, where $l$ is the agent's location and $w$ is the worm's location.

*Learning a Correct Model:* Although the theorem of convergence of PGRD relies on the agent having a fixed model, the algorithm itself is readily applied to the case of learning a model online. In this experiment, the agent's model $T$ is learned online based on empirical transition probabilities between observations (recall this is a fully observable environment). Let $n_{o,a,o'}$ be the number of times that $o'$ was reached after taking action $a$ after observing $o$. The agent models the probability of seeing $o'$ as $T(o'|o, a) = \frac{n_{o,a,o'}}{\sum_{o'} n_{o,a,o'}}$.

*Reward Parameterizations:* Recall that $R(i, o, a; \theta) = \theta^\mathsf{T} \phi(i, o, a)$, for some $\phi(i, o, a)$. (1) In the *observation-action parameterization*, $\phi(i, o, a)$ is a binary feature vector with one binary feature for each observation-action pair—internal state is ignored. This is effectively a table representation over all reward functions indexed by $(o, a)$. As shown in Theorem 3, the observation-action feature representation is capable of producing arbitrary policies over the observations. In large problems, such a parameterization would not be feasible. (2) The *recency parameterization* is a more compact representation which uses features that rely on the history of observations. The feature vector is $\phi(i, o, a) = [R_\mathcal{O}(o, a), 1, \phi_{c_l}(l, i), \phi_{c_{l,a}}(l, a, i)]$, where $R_\mathcal{O}(o, a)$ is the objective reward function defined as above. The feature $\phi_{c_l}(l) = 1 - 1/c(l, i)$, where $c(l, i)$ is the number of time steps since the agent has visited location $l$, as represented in the agent's internal state $i$. Its value is normalized to the range $[0, 1)$ and is high when the agent has not been to location $l$ recently. The feature $\phi_{c_{l,a}}(l, a, i) = 1 - 1/c(l, a, i)$ is similarly defined with respect to the time since the agent has taken action $a$ in location $l$. Features based on recency counts encourage persistent exploration [21, 18].

*Results & Discussion:* Figure 2(b) and Figure 2(c) present results for agents that use the observation-action parameterization and the recency parameterization of the reward function respectively. The horizontal axis is the number of time steps of experience. The vertical axis is the objective return, i.e., the average objective reward per time step. Each curve is an average over 130 trials. The values of $d$ and the associated optimal algorithm parameters for each curve are noted in the figures. First, note that with $d = 6$, the agent is unbounded, because food is never more than 6 steps away. Therefore, the agent does not benefit from adapting the reward function parameters (given that we initialize to the objective reward function). Indeed, the $d = 6, \alpha = 0$ agent performs as well as the best reward-optimizing agent. The performance for $d = 6$ improves with experience because the model improves with experience (and thus from the curves it is seen that the model gets quite accurate in about 1500 time steps). The largest objective return obtained for $d = 6$ is also the best objective return that can be obtained for any value of $d$.

Several results can be observed in both Figures 2(b) and (c). 1) Each curve that uses $\alpha > 0$ (solid lines) improves with experience. This is a demonstration of our primary contribution, that PGRD is able to effectively improve the reward function with experience. That the improvement over time is not just due to model learning is seen in the fact that for each value of $d < 6$ the curve for $\alpha > 0$ (solid-line) which adapts the reward parameters does significantly better than the corresponding curve for $\alpha = 0$ (dashed-line); the $\alpha = 0$ agents still learn the model. 2) For both $\alpha = 0$ and $\alpha > 0$ agents, the objective return obtained by agents with equivalent amounts of experience increases monotonically as $d$ is increased (though to maintain readability we only show selected values of $d$ in each figure). This demonstrates our secondary contribution, that the ability to plan in PGRD significantly improves performance over standard OLPOMDP (PGRD with $d = 0$).

There are also some interesting differences between the results for the two different reward function parameterizations. With the observation-action parameterization, we noted that there always exists a setting of $\theta$ for all $d$ that will yield optimal objective return. This is seen in Figure 2(b) in that all solid-line curves approach optimal objective return. In contrast, the more compact recency reward parameterization does not afford this guarantee and indeed for small values of $d$ ($< 3$), the solid-line curves in Figure 2(c) converge to less than optimal objective return. Notably, OLPOMDP ($d = 0$) does not perform well with this feature set. On the other hand, for planning depths $3 \leq d < 6$, the PGRD agents with the recency parameterization achieve optimal objective return faster than the corresponding PGRD agent with the observation-action parameterization. Finally, we note that this experiment validates our claim that PGRD can improve reward functions that depend on history.

**Experiment 2: A fully observable environment and poor given model.** Our theoretical analysis showed that PGRD with an incorrect model and the observation–action reward parameterization should (modulo local maxima issues) do just as well asymptotically as it would with a correct model. Here we illustrate this theoretical result empirically on the same foraging domain and objective reward function used in Experiment 1. We also test our hypothesis that a poor model should slow down the rate of learning relative to a correct model.

*Poor Model:* We gave the agents a fixed incorrect model of the foraging environment that assumes there are no internal walls separating the 3 corridors.
*Reward Parameterization:* We used the observation–action reward parameterization. With a poor model it is no longer interesting to initialize $\theta$ so that the initial reward function is the objective reward function because even for $d = 6$ such an agent would do poorly. Furthermore, we found that this initialization leads to excessively bad exploration and therefore poor learning of how to modify the reward. Thus, we initialize $\theta$ to uniform random values near 0, in the range $(-10^{-3}, 10^{-3})$.

*Results:* Figure 3(a) plots the objective return as a function of number of steps of experience. Each curve is an average over 36 trials. As hypothesized, the bad model slows learning by a factor of more than 10 (notice the difference in the x-axis scales from those in Figure 2). Here, deeper planning results in slower learning and indeed the $d = 0$ agent that does not use the model at all learns the fastest. However, also as hypothesized, because they used the expressive observation–action parameterization, agents of all planning depths mitigate the damage caused by the poor model and eventually converge to the optimal objective return.

**Experiment 3: Partially observable foraging world.** Here we evaluate PGRD's ability to learn in a partially observable version of the foraging domain. In addition, the agents learn a model under the erroneous (and computationally convenient) assumption that the domain is fully observable.

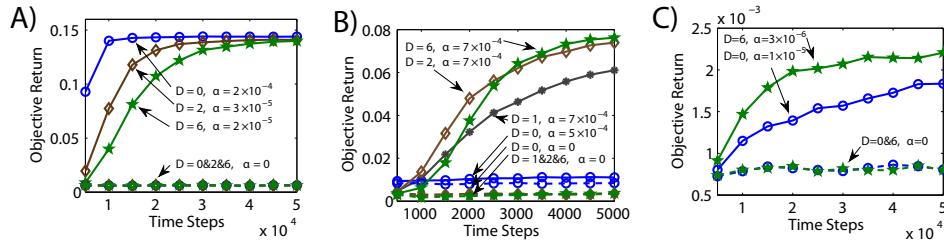

Figure 3: A) Performance of PGRD with a poor model, B) Performance of PGRD in a partially observable world with recency reward features, C) Performance of PGRD in Acrobot

*Partial Observation:* Instead of viewing the location of the worm at all times, the agent can now only see the worm when it is colocated with it: its observation is $o = (l, f)$, where $f$ indicates whether the agent is colocated with the food.

*Learning an Incorrect Model:* The model is learned just as in Experiment 1. Because of the erroneous full observability assumption, the model will hallucinate about worms at all the corridor ends based on the empirical frequency of having encountered them there.

*Reward Parameterization:* We used the recency parameterization; due to the partial observability, agents with the observation–action feature set perform poorly in this environment. The parameters $\theta$ are initialized such that the initial reward function equals the objective reward function.

*Results & Discussion:* Figure 3(b) plots the mean of 260 trials. As seen in the solid-line curves, PGRD improves the objective return at all depths (only a small amount for $d = 0$ and significantly more for $d > 0$). In fact, agents which don't adapt the reward are hurt by planning (relative to $d = 0$). This experiment demonstrates that the combination of planning and reward improvement can be beneficial even when the model is erroneous. Because of the partial observability, optimal behavior in this environment achieves less objective return than in Experiment 1.

**Experiment 4: Acrobot.** In this experiment we test PGRD in the Acrobot environment [22], a common benchmark task in the RL literature and one that has previously been used in the testing of policy gradient approaches [23]. This experiment demonstrates PGRD in an environment in which an agent must be limited due to the size of the state space and further demonstrates that adding model-based planning to policy gradient approaches can improve performance.

*Domain:* The version of Acrobot we use is as specified by Sutton and Barto [22]. It is a two-link robot arm in which the position of one shoulder-joint is fixed and the agent's control is limited to 3 actions which apply torque to the elbow-joint.

*Observation:* The fully-observable state space is 4 dimensional, with two joint angles $\psi_1$ and $\psi_2$, and two joint velocities $\dot{\psi}_1$ and $\dot{\psi}_2$.

*Objective Reward:* The designer receives an objective reward of $1.0$ when the tip is one arm's length above the fixed shoulder-joint, after which the bot is reset to its initial resting position.

*Model:* We provide the agent with a perfect model of the environment. Because the environment is continuous, value iteration is intractable, and computational limitations prevent planning deep enough to compute the optimal action in any state. The feature vector contains 13 entries. One feature corresponds to the objective reward signal. For each action, there are 5 features corresponding to each of the state features plus an additional feature representing the height of the tip: $\phi(i, o, a) = [R_{\mathcal{O}}(o), \{\psi_1(o), \psi_2(o), \dot{\psi}_1(o), \dot{\psi}_2(o), h(o)\}_a]$. The height feature has been used in previous work as an alternative definition of objective reward [23].

*Results & Discussion:* We plot the mean of 80 trials in Figure 3(c). Agents that use the fixed ($\alpha = 0$) objective reward function with bounded-depth planning perform according to the bottom two curves. Allowing PGRD and OLPOMDP to adapt the parameters $\theta$ leads to improved objective return, as seen in the top two curves in Figure 3(c). Finally, the PGRD $d = 6$ agent outperforms the standard OLPOMDP agent (PGRD with $d = 0$), further demonstrating that PGRD outperforms OLPOMDP.

**Overall Conclusion:** We developed PGRD, a new method for approximately solving the optimal reward problem in bounded planning agents that can be applied in an online setting. We showed that PGRD is a generalization of OLPOMDP and demonstrated that it both improves reward functions in limited agents and outperforms the model-free OLPOMDP approach.

# References

[1] Douglas Aberdeen and Jonathan Baxter. Scalable Internal-State Policy-Gradient Methods for POMDPs. *Proceedings of the Nineteenth International Conference on Machine Learning*, 2002.

[2] Peter L. Bartlett and Jonathan Baxter. Stochastic optimization of controlled partially observable Markov decision processes. In *Proceedings of the 39th IEEE Conference on Decision and Control*, 2000.

[3] Jonathan Baxter, Peter L. Bartlett, and Lex Weaver. Experiments with Infinite-Horizon, Policy-Gradient Estimation, 2001.

[4] Shalabh Bhatnagar, Richard S. Sutton, M Ghavamzadeh, and Mark Lee. Natural actor-critic algorithms. *Automatica*, 2009.

[5] Ronen I. Brafman and Moshe Tennenholtz. R-MAX - A General Polynomial Time Algorithm for Near-Optimal Reinforcement Learning. *Journal of Machine Learning Research*, 3:213–231, 2001.

[6] S. Elfwing, Eiji Uchibe, K. Doya, and H. I. Christensen. Co-evolution of Shaping Rewards and Meta-Parameters in Reinforcement Learning. *Adaptive Behavior*, 16(6):400–412, 2008.

[7] J. Zico Kolter and Andrew Y. Ng. Near-Bayesian exploration in polynomial time. In *Proceedings of the 26th International Conference on Machine Learning*, pages 513–520, 2009.

[8] Harold J. Kushner and G. George Yin. *Stochastic Approximation and Recursive Algorithms and Applications*. Springer, 2nd edition, 2010.

[9] Çetin Meriçli, Tekin Meriçli, and H. Levent Akin. A Reward Function Generation Method Using Genetic Algorithms : A Robot Soccer Case Study (Extended Abstract). In *Proc. of the 9th Int. Conf. on Autonomous Agents and Multiagent Systems (AAMAS 2010)*, number 2, pages 1513–1514, 2010.

[10] Gergely Neu and Csaba Szepesvári. Apprenticeship learning using inverse reinforcement learning and gradient methods. In *Proceedings of the 23rd Conference on Uncertainty in Artificial Intelligence*, pages 295–302, 2007.

[11] Andrew Y. Ng, Stuart J. Russell, and D. Harada. Policy invariance under reward transformations: Theory and application to reward shaping. In *Proceedings of the 16th International Conference on Machine Learning*, pages 278–287, 1999.

[12] Scott Niekum, Andrew G. Barto, and Lee Spector. Genetic Programming for Reward Function Search. *IEEE Transactions on Autonomous Mental Development*, 2(2):83–90, 2010.

[13] Pierre-Yves Oudeyer, Frederic Kaplan, and Verena V. Hafner. Intrinsic Motivation Systems for Autonomous Mental Development. *IEEE Transactions on Evolutionary Computation*, 11(2):265–286, April 2007.

[14] Jürgen Schmidhuber. Curious model-building control systems. In *IEEE International Joint Conference on Neural Networks*, pages 1458–1463, 1991.

[15] Satinder Singh, Andrew G. Barto, and Nuttapong Chentanez. Intrinsically Motivated Reinforcement Learning. In *Proceedings of Advances in Neural Information Processing Systems 17 (NIPS)*, pages 1281–1288, 2005.

[16] Satinder Singh, Richard L. Lewis, and Andrew G. Barto. Where Do Rewards Come From? In *Proceedings of the Annual Conference of the Cognitive Science Society*, pages 2601–2606, 2009.

[17] Satinder Singh, Richard L. Lewis, Andrew G. Barto, and Jonathan Sorg. Intrinsically Motivated Reinforcement Learning: An Evolutionary Perspective. *IEEE Transations on Autonomous Mental Development*, 2(2):70–82, 2010.

[18] Jonathan Sorg, Satinder Singh, and Richard L. Lewis. Internal Rewards Mitigate Agent Boundedness. In *Proceedings of the 27th International Conference on Machine Learning*, 2010.

[19] Jonathan Sorg, Satinder Singh, and Richard L. Lewis. Variance-Based Rewards for Approximate Bayesian Reinforcement Learning. In *Proceedings of the 26th Conference on Uncertainty in Artificial Intelligence*, 2010.

[20] Alexander L. Strehl and Michael L. Littman. An analysis of model-based Interval Estimation for Markov Decision Processes. *Journal of Computer and System Sciences*, 74(8):1309–1331, 2008.

[21] Richard S. Sutton. Integrated Architectures for Learning, Planning, and Reacting Based on Approximating Dynamic Programming. In *The Seventh International Conference on Machine Learning*, pages 216–224. 1990.

[22] Richard S. Sutton and Andrew G. Barto. *Reinforcement Learning: An Introduction*. The MIT Press, 1998.

[23] Lex Weaver and Nigel Tao. The Optimal Reward Baseline for Gradient-Based Reinforcement Learning. In *Proceedings of the 17th Conference on Uncertainty in Artificial Intelligence*, pages 538–545. 2001.

